# Adaptive Embedded Subgraph Algorithms using Walk-Sum Analysis

**Venkat Chandrasekaran, Jason K. Johnson, and Alan S. Willsky**
Department of Electrical Engineering and Computer Science
Massachusetts Institute of Technology
venkatc@mit.edu, jasonj@mit.edu, willsky@mit.edu

## Abstract

We consider the estimation problem in Gaussian graphical models with arbitrary structure. We analyze the Embedded Trees algorithm, which solves a sequence of problems on tractable subgraphs thereby leading to the solution of the estimation problem on an intractable graph. Our analysis is based on the recently developed walk-sum interpretation of Gaussian estimation. We show that non-stationary iterations of the Embedded Trees algorithm using *any* sequence of subgraphs converge in walk-summable models. Based on walk-sum calculations, we develop adaptive methods that optimize the choice of subgraphs used at each iteration with a view to achieving maximum reduction in error. These adaptive procedures provide a significant speedup in convergence over stationary iterative methods, and also appear to converge in a larger class of models.

## 1 Introduction

Stochastic processes defined on graphs offer a compact representation for the Markov structure in a large collection of random variables. We consider the class of Gaussian processes defined on graphs, or Gaussian graphical models, which are used to model natural phenomena in many large-scale applications [1, 2]. In such models, the estimation problem can be solved by directly inverting the information matrix. However, the resulting complexity is cubic in the number of variables, thus being prohibitively complex in applications involving hundreds of thousands of variables. Algorithms such as Belief Propagation and the junction-tree method are effective for computing exact estimates in graphical models that are tree-structured or have low treewidth [3], but for graphs with high treewidth the junction-tree approach is intractable.

We describe a rich class of iterative algorithms for estimation in Gaussian graphical models with arbitrary structure. Specifically, we discuss the Embedded Trees (ET) iteration [4] that solves a sequence of estimation problems on trees, or more generally tractable subgraphs, leading to the solution of the original problem on the intractable graph. We analyze non-stationary iterations of the ET algorithm that perform inference calculations on an *arbitrary* sequence of subgraphs. Our analysis is based on the recently developed walk-sum interpretation of inference in Gaussian graphical models [5]. We show that in the broad class of so-called walk-summable models, the ET algorithm converges for *any* arbitrary sequence of subgraphs used. The walk-summability of a model is easily tested [5, 6], thus providing a simple sufficient condition for the convergence of such non-stationary algorithms. Previous convergence results [6, 7] analyzed stationary or "cyclo-stationary" iterations that use the same subgraph at each iteration or cycle through a fixed sequence of subgraphs. The focus of this paper is on analyzing, and developing algorithms based on, arbitrary non-stationary iterations that use any (non-cyclic) sequence of subgraphs, and the recently developed concept of walk-sums appears to be critical to this analysis.

Given this great flexibility in choosing successive iterative steps, we develop algorithms that adaptively optimize the choice of subgraphs to achieve maximum reduction in error. These algorithms take advantage of walk-sum calculations, which are useful in showing that our methods minimize an upper-bound on the error at each iteration. We develop two procedures to adaptively choose subgraphs. The first method finds the best tree at each iteration by solving an appropriately formulated maximum-weight spanning tree problem, with the weight of each edge being a function of the partial correlation coefficient of the edge and the residual errors at the nodes that compose the edge. The second method, building on this first method, adds extra edges in a greedy manner to the tree resulting from the first method to form a thin hypertree. Simulation results demonstrate that these non-stationary algorithms provide a significant speedup in convergence over stationary and cyclic iterative methods. Since the class of walk-summable models is broad (including attractive models, diagonally dominant models, and so-called pairwise-normalizable models), our methods provide a convergent, computationally attractive method for inference. We also provide empirical evidence to show that our adaptive methods (with a minor modification) converge in many non-walk-summable models when stationary iterations diverge. The estimation problem in Gaussian graphical models involves solving a linear system with a sparse, symmetric, positive-definite matrix. Such systems are commonly encountered in other areas of machine learning and signal processing as well [8, 9]. Therefore, our methods are broadly applicable beyond estimation in Gaussian models.

Some of the results presented here appear in more detail in a longer paper [10], which provides complete proofs as well as a detailed description of *walk-sum diagrams* that give a graphical interpretation of our algorithms (we show an example in this paper). The report also considers problems involving communication "failure" between nodes for distributed sensor network applications.

## 2  Background

Let $\mathcal{G} = (V, \mathcal{E})$ be a graph with vertices $V$, and edges $\mathcal{E} \subset \binom{V}{2}$ that link pairs of vertices together. Here, $\binom{V}{2}$ represents the set of all unordered pairs of vertices. Consider a Gaussian distribution in *information form* [5] $p(x) \propto \exp\{-\frac{1}{2}x^T Jx + h^T x\}$, where $J^{-1}$ is the covariance matrix and $J^{-1}h$ is the mean. The matrix $J$, also called the *information matrix*, is sparse according to graph $\mathcal{G}$, i.e. $J_{s,t} = J_{t,s} = 0$ if and only if $\{s, t\} \notin \mathcal{E}$. Thus, $\mathcal{G}$ represents the graph with respect to which $p(x)$ is Markov, i.e. $p(x)$ satisfies the conditional independencies implied by the separators of $\mathcal{G}$. The Gaussian mean estimation problem reduces to solving the following linear system of equations:

$$Jx = h, \tag{1}$$

where $x$ is the mean vector. Convergent iterations that compute the mean can also be used in turn to compute variances using a variety of methods [4, 11]. Thus, we focus on the problem of estimating the mean at each node. Throughout the rest of this paper, we assume that $J$ is normalized to have 1's along the diagonal.[1] Such a re-scaling does not affect the convergence results in this paper, and our analysis and algorithms can be easily generalized to the un-normalized case [10].

### 2.1  Walk-sums

We give a brief overview of the walk-sum framework developed in [5]. Let $J = I - R$. The off-diagonal entries of the matrix $R$ have the same sparsity structure as that of $J$, and consequently that of the graph $\mathcal{G}$. For Gaussian processes defined on graphs, element $R_{s,t}$ corresponds to the conditional correlation coefficient between the variables at vertices $s$ and $t$ conditioned on knowledge of all the other variables (also known as the *partial correlation coefficient* [5]). A *walk* is a sequence of vertices $\{w_i\}_{i=0}^{\ell}$ such that each step $\{w_i, w_{i+1}\} \in \mathcal{E}, 0 \leq i \leq \ell - 1$, with no restriction on crossing the same vertex or traversing the same edge multiple times. The *weight* of a walk is the product of the edge-wise partial correlation coefficients of the edges composing the walk: $\phi(w) \triangleq \prod_{i=0}^{\ell-1} R_{w_i, w_{i+1}}$. We then have that $(R^{\ell})_{s,t}$ is the sum of the weights of all length-$\ell$ walks from $s$ to $t$ in $\mathcal{G}$. With this point of view, we can interpret $J^{-1}$ as follows:

$$(J^{-1})_{s,t} = ((I - R)^{-1})_{s,t} = \sum_{\ell=0}^{\infty} (R^{\ell})_{s,t} = \sum_{\ell=0}^{\infty} \phi(s \xrightarrow{\ell} t), \tag{2}$$

where $\phi(s \xrightarrow{\ell} t)$ represents the sum of the weights of all the length-$\ell$ walks from $s$ to $t$ (the set of all such walks is finite). Thus, $(J^{-1})_{s,t}$ is the length-ordered sum over all walks in $\mathcal{G}$ from $s$ to $t$. This, however, is a very specific way to compute the inverse that converges if the spectral radius $\varrho(R) < 1$. Other algorithms may compute walks according to different orders (rather than length-based orders). To analyze arbitrary algorithms that submit to a walk-sum interpretation, the following concept of *walk-summability* was developed in [5]. A model is said to be *walk-summable* if for each pair of vertices $s, t \in V$, the absolute sum over all walks from $s$ to $t$ in $\mathcal{G}$ converges:

$$\bar{\phi}(s \to t) \triangleq \sum_{w \in \mathcal{W}(s \to t)} |\phi(w)| < \infty. \tag{3}$$

Here, $\mathcal{W}(s \to t)$ represents the set of all walks from $s$ to $t$, and $\bar{\phi}(s \to t)$ denotes the absolute walk-sum[2] over this set. Based on the absolute convergence condition, walk-summability implies that walk-sums over a countable set of walks in $\mathcal{G}$ can be computed in any order. As a result, we have the following interpretation in walk-summable models:

$$(J^{-1})_{s,t} = \phi(s \to t), \tag{4}$$

$$x_t = (J^{-1}h)_t = \sum_{s \in V} h_s \phi(s \to t) \triangleq \phi(h; * \to t), \tag{5}$$

where the wildcard character $*$ denotes a union over all vertices in $V$, and $\phi(h; \mathcal{W})$ denotes a re-weighting of each walk in $\mathcal{W}$ by the corresponding $h$ value at the starting node. Note that in (4) we relax the constraint that the sum is ordered by length, and do not explicitly specify an ordering on the walks (such as in (2)). In words, $(J^{-1})_{s,t}$ is the walk-sum over the set of all walks from $s$ to $t$, and $x_t$ is the walk-sum over all walks ending at $t$, re-weighted by $h$.

As shown in [5], the walk-summability of a model is equivalent to $\varrho(\bar{R}) < 1$, where $\bar{R}$ denotes the matrix of the absolute values of the elements of $R$. Also, a broad class of models are walk-summable, including diagonally-dominant models, so-called pairwise normalizable models, and models for which the underlying graph $\mathcal{G}$ is non-frustrated, i.e. each cycle has an even number of negative partial correlation coefficients. Walk-summability implies that a model is valid, i.e. has positive-definite information/covariance.

**Concatenation of walks** We briefly describe the concatenation operation for walks and walk-sets, which plays a key role in walk-sum analysis. Let $u = u_0 \cdots u_{end}$ and $v = v_{start} v_1 \cdots v_{\ell(v)}$ be walks with $u_{end} = v_{start}$. The concatenation of these walks is defined to be $u \cdot v \triangleq u_0 \cdots u_{end} v_1 \cdots v_{\ell(v)}$. Now consider a walk-set $\mathcal{U}$ with all walks ending at $u_{end}$ and another walk-set $\mathcal{V}$ with all walks beginning at $v_{start}$. If $u_{end} = v_{start}$, then the concatenation of $\mathcal{U}$ and $\mathcal{V}$ is defined:

$$\mathcal{U} \otimes \mathcal{V} \triangleq \{u \cdot v : u \in \mathcal{U}, v \in \mathcal{V}\}.$$

## 2.2 Embedded Trees algorithm

We describe the Embedded Trees iteration that performs a sequence of updates on trees, or more generally tractable subgraphs, leading to the solution of (1) on an intractable graph. Each iteration involves an inference calculation on a subgraph of *all* the variables $V$. Let $(V, \mathcal{S})$ be some subgraph of $\mathcal{G}$, i.e. $\mathcal{S} \subset \mathcal{E}$ (see examples in Figure 1). Let $J$ be split according to $\mathcal{S}$ as $J = J_{\mathcal{S}} - K_{\mathcal{S}}$, so that the entries of $J$ corresponding to edges in $\mathcal{S}$ are assigned to $J_{\mathcal{S}}$, and those corresponding to $\mathcal{E} \backslash \mathcal{S}$ are part of $K_{\mathcal{S}}$. The diagonal entries of $J$ are all part of $J_{\mathcal{S}}$; thus, $K_{\mathcal{S}}$ has zeroes along the diagonal.[3] Based on this splitting, we can transform (1) to $J_{\mathcal{S}} x = K_{\mathcal{S}} x + h$, which suggests a natural recursion: $J_{\mathcal{S}} \widehat{x}^{(n)} = K_{\mathcal{S}} \widehat{x}^{(n-1)} + h$. If $J_{\mathcal{S}}$ is invertible, and it is tractable to apply $J_{\mathcal{S}}^{-1}$ to a vector, then ET offers an effective method to solve (1) (assuming $\varrho(J_{\mathcal{S}}^{-1} K_{\mathcal{S}}) < 1$). If the subgraph used changes with each iteration, then we obtain the following *non-stationary* ET iteration:

$$\widehat{x}^{(n)} = J_{\mathcal{S}_n}^{-1}(K_{\mathcal{S}_n} \widehat{x}^{(n-1)} + h), \tag{6}$$

where $\{\mathcal{S}_n\}_{n=1}^{\infty}$ is any arbitrary sequence of subgraphs. An important degree of freedom is the choice of the subgraph $\mathcal{S}_n$ at iteration $n$, which forms the focus of Section 4 of this paper. In [10] we also consider a more general class of algorithms that update *subsets* of variables at each iteration.

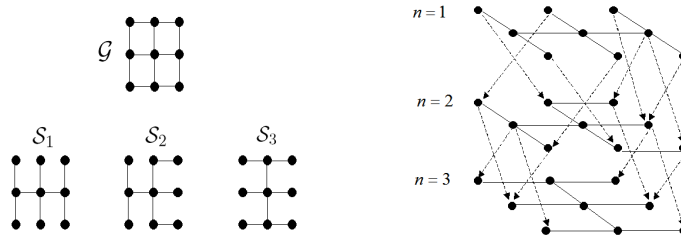

Figure 1: (Left) $\mathcal{G}$ and three embedded trees $\mathcal{S}_1, \mathcal{S}_2, \mathcal{S}_3$; (Right) Corresponding walk-sum diagram.

# 3 Walk-Sum Analysis and Convergence of the Embedded Trees algorithm

In this section, we provide a walk-sum interpretation for the ET algorithm. Using this analysis, we show that the non-stationary ET iteration (6) converges in walk-summable models for an arbitrary choice of subgraphs $\{\mathcal{S}_n\}_{n=1}^{\infty}$. Before proceeding with the analysis, we point out that one potential complication with the ET algorithm is that the matrix $J_{\mathcal{S}}$ corresponding to some subgraph $\mathcal{S}$ may be indefinite or singular, even if the original model $J$ is positive-definite. Importantly, such a problem never arises in walk-summable models with $J_{\mathcal{S}}$ being positive-definite for any subgraph $\mathcal{S}$ if $J$ is walk-summable. This is easily seen because walks in the subgraph $\mathcal{S}$ are a subset of the walks in $\mathcal{G}$, and thus if absolute walk-sums in $\mathcal{G}$ are well-defined, then so are absolute walk-sums in $\mathcal{S}$. Therefore, $J_{\mathcal{S}}$ is walk-summable, and hence, positive-definite.

Consider the following recursively defined set of walks for $s, t \in V$:

$$\mathcal{W}_n(s \rightarrow t) = \left[ \cup_{u,v \in V} \mathcal{W}_{n-1}(s \rightarrow u) \otimes \mathcal{W}(u \xrightarrow{\mathcal{E} \backslash \mathcal{S}_n(1)} v) \otimes \mathcal{W}(v \xrightarrow{\mathcal{S}_n} t) \right] \bigcup \mathcal{W}(s \xrightarrow{\mathcal{S}_n} t)$$

$$= \mathcal{W}_{n-1}(s \rightarrow *) \otimes \mathcal{W}(* \xrightarrow{\mathcal{E} \backslash \mathcal{S}_n(1)} \bullet) \otimes \mathcal{W}(\bullet \xrightarrow{\mathcal{S}_n} t) \bigcup \mathcal{W}(s \xrightarrow{\mathcal{S}_n} t), \tag{7}$$

with $\mathcal{W}_0(s \rightarrow t) = \emptyset$. Here, $*$ and $\bullet$ are used as wildcard characters (a union over all elements in $V$), and $\otimes$ denotes concatenation of walk-sets as described previously. The set $\mathcal{W}_{n-1}(s \rightarrow *)$ denotes walks that start at node $s$ computed at the previous iteration. The middle term $\mathcal{W}(* \xrightarrow{\mathcal{E} \backslash \mathcal{S}_n(1)} \bullet)$ denotes a length-1 walk (called a *hop*) across an edge in $\mathcal{E} \backslash \mathcal{S}_n$. Finally, $\mathcal{W}(\bullet \xrightarrow{\mathcal{S}_n} t)$ denotes walks in $\mathcal{S}_n$ that end at node $t$. Thus, the first term in (7) refers to previously computed walks starting at $s$, which hop across an edge in $\mathcal{E} \backslash \mathcal{S}_n$, and then finally propagate only in $\mathcal{S}_n$ (ending at $t$). The second term $\mathcal{W}(s \xrightarrow{\mathcal{S}_n} t)$ denotes walks from $s$ to $t$ that only live within $\mathcal{S}_n$. The following proposition (proved in [10]) shows that the walks contained in these walk-sets are precisely those computed by the ET algorithm at iteration $n$. For simplicity, we denote $\phi(\mathcal{W}_n(s \rightarrow t))$ by $\phi_n(s \rightarrow t)$.

**Proposition 1** *Let $\widehat{x}^{(n)}$ be the estimate at iteration $n$ in the ET algorithm (6) with initial guess $\widehat{x}^{(0)} = 0$. Then, $\widehat{x}_t^{(n)} = \phi_n(h; * \rightarrow t) = \sum_{s \in V} h_s \phi_n(s \rightarrow t)$ in walk-summable models.*

We note that the classic Gauss-Jacobi algorithm [6], a stationary iteration with $J_{\mathcal{S}} = I$ and $K_{\mathcal{S}} = R$, can be interpreted as a walk-sum algorithm: $\widehat{x}_t^{(n)}$ in this method computes all walks up to length $n$ ending at $t$. Figure 1 gives an example of a *walk-sum diagram*, which provides a graphical representation of the walks accumulated by the walk-sets (7). The diagram is the three-level graph on the right, and corresponds to an ET iteration based on the subgraphs $\mathcal{S}_1, \mathcal{S}_2, \mathcal{S}_3$ of the $3 \times 3$ grid $\mathcal{G}$ (on the left). Each level $n$ in the diagram consists of the subgraph $\mathcal{S}_n$ used at iteration $n$ (solid edges), and information from the previous level (iteration) $n - 1$ is transmitted through the dashed edges in $\mathcal{E} \backslash \mathcal{S}_n$. The directed nature of these dashed edges is critical as they capture the one-directional flow of computations from iteration to iteration, while the undirected edges within each level capture the inference computation at each iteration. Consider a node $v$ at level $n$ of the diagram. Walks in the diagram that start at any node and end at $v$ at level $n$, re-weighted by $h$, are exactly the walks computed by the ET algorithm in $\widehat{x}_v^{(n)}$. For more examples of such diagrams, see [10].

Given this walk-sum interpretation of the ET algorithm, we can analyze the walk-sets (7) to prove the convergence of ET in walk-summable models by showing that the walk-sets eventually contain all the walks required for the computation of $J^{-1}h$ in (5). We have the following convergence theorem for which we only provide a brief sketch of the complete proof [10].

**Theroem 1** *Let $\widehat{x}^{(n)}$ be the estimate at iteration $n$ in the ET algorithm (6) with initial guess $\widehat{x}^{(0)} = 0$. Then, $\widehat{x}^{(n)} \to J^{-1}h$ element-wise as $n \to \infty$ in walk-summable models.*

**Proof outline**: Proving this statement is done in the following stages.

*Validity*: The walks in $\mathcal{W}_n$ are valid walks in $\mathcal{G}$, i.e. $\mathcal{W}_n(s \to t) \subseteq \mathcal{W}(s \to t)$.

*Nesting*: The walk-sets $\mathcal{W}_n(s \to t)$ are nested, i.e. $\mathcal{W}_{n-1}(s \to t) \subseteq \mathcal{W}_n(s \to t), \forall n$.

*Completeness*: Let $w \in \mathcal{W}(s \to t)$. There exists an $N > 0$ such that $w \in \mathcal{W}_N(s \to t)$. Using the nesting property, we conclude that for all $n \geq N$, $w \in \mathcal{W}_n(s \to t)$.

These steps combined together allow us to conclude that $\phi_n(s \to t) \to \phi(s \to t)$ as $n \to \infty$. This conclusion relies on the fact that $\phi(\mathcal{W}_n) \to \phi(\cup_n \mathcal{W}_n)$ as $n \to \infty$ for a sequence of nested walk-sets $\mathcal{W}_{n-1} \subseteq \mathcal{W}_n$ in walk-summable models, which is a consequence of the sum-partition theorem for absolutely summable series [5, 10, 12]. Given the walk-sum interpretation from Proposition 1, one can check that $\widehat{x}^{(n)} \to J^{-1}h$ element-wise as $n \to \infty$. $\square$

Thus, the ET algorithm converges to the correct solution of (1) in walk-summable models for any sequence of subgraphs with $\widehat{x}^{(0)} = 0$. It is then straightforward to show that convergence can be achieved for any initial guess [10]. Note that we have taken advantage of the absolute convergence property in walk-summable models (3) by not focusing on the order in which walks are computed, but only *that* they are eventually computed. In [10], we prove that walk-summability is also a *necessary* condition for the complete flexibility in the choice of subgraphs — there exists at least one sequence of subgraphs that results in a divergent ET iteration in non-walk-summable models.

## 4   Adaptive algorithms

Let $e^{(n)} = x - \widehat{x}^{(n)}$ be the *error* at iteration $n$ and let $h^{(n)} = Je^{(n)} = h - J\widehat{x}^{(n)}$ be the corresponding *residual error* (which is tractable to compute). We begin by describing an algorithm to choose the "next-best" tree $\mathcal{S}_n$ in the ET iteration (6). The error at iteration $n$ can be re-written as follows:

$$e^{(n)} = (J^{-1} - J_{\mathcal{S}_n}^{-1})h^{(n-1)}.$$

Thus, we have the walk-sum interpretation $e_t^{(n)} = \phi(h^{(n-1)}; * \xrightarrow{\mathcal{G}\backslash\mathcal{S}_n} t)$, where $\mathcal{G}\backslash\mathcal{S}_n$ denotes walks that do not live entirely within $\mathcal{S}_n$. Using this expression for the error, we have the following bound that is tight for attractive models ($R_{s,t} \geq 0$ for all $s, t \in V$) and non-negative $h^{(n-1)}$:

$$
\begin{aligned}
\|e^{(n)}\|_{\ell_1} &= \sum_{t \in V} |\phi(h^{(n-1)}; * \xrightarrow{\mathcal{G}\backslash\mathcal{S}_n} t)| \\
&\leq \bar{\phi}(|h^{(n-1)}|; \mathcal{G}\backslash\mathcal{S}_n) \\
&= \bar{\phi}(|h^{(n-1)}|; \mathcal{G}) - \bar{\phi}(|h^{(n-1)}|; \mathcal{S}_n).
\end{aligned}
\tag{8}
$$

Hence, minimizing the error at iteration $n$ corresponds to finding the tree $\mathcal{S}_n$ that *maximizes* the second term $\bar{\phi}(|h^{(n-1)}|; \mathcal{S}_n)$. This leads us to the following *maximum walk-sum tree* problem:

$$\arg \max_{\mathcal{S}_n \text{ a tree}} \bar{\phi}(|h^{(n-1)}|; \mathcal{S}_n) \tag{9}$$

Finding the optimal such tree is combinatorially complex. Therefore, we develop a relaxation that minimizes a looser upper bound than (8). Specifically, consider an edge $\{u, v\}$ and all the walks that live on this single edge $\mathcal{W}(\{u, v\}) = \{uv, vu, uvu, vuv, uvuv, vuvu, \dots\}$. One can check that the contribution based on these single-edge walks can be computed as:

$$\sigma_{u,v} = \sum_{w \in \mathcal{W}(\{u,v\})} \bar{\phi}(|h^{(n-1)}|; w) = \left(|h_u^{(n-1)}| + |h_v^{(n-1)}|\right) \frac{|R_{u,v}|}{1 - |R_{u,v}|}. \tag{10}$$

This weight provides a measure of the error-reduction capacity of edge $\{u, v\}$ by itself at iteration $n$. These single-edge walks for edges in $\mathcal{S}_n$ are a subset of all the walks in $\mathcal{S}_n$, and consequently provide a lower-bound on $\bar{\phi}(|h^{(n-1)}|; \mathcal{S}_n)$. Therefore, the maximization

$$\arg \max_{\mathcal{S}_n \text{ a tree}} \sum_{\{u,v\} \in \mathcal{S}_n} \sigma_{u,v} \tag{11}$$

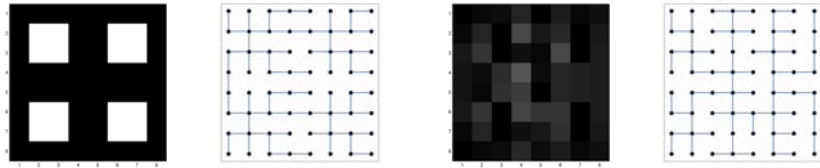

Figure 2: Grayscale images of residual errors in an $8 \times 8$ grid at successive iterations, and corresponding trees chosen by adaptive method.

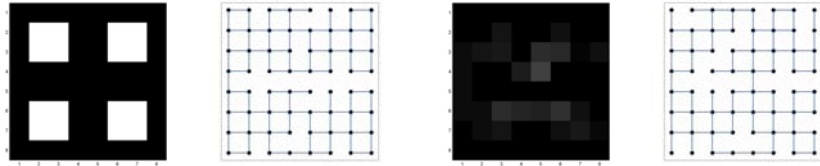

Figure 3: Grayscale images of residual errors in an $8 \times 8$ grid at successive iterations, and corresponding hypertrees chosen by adaptive method.

is equivalent to minimizing a looser upper-bound than (8). This relaxed problem can be solved efficiently using a maximum-weight spanning tree algorithm that has complexity $\mathcal{O}(|\mathcal{E}| \log \log |V|)$ for sparse graphs [13].

Given the maximum-weight spanning tree of the graph, a natural extension is to build a *thin hypertree* by adding extra "strong" edges to the tree, subject to the constraint that the resulting graph has low treewidth. Unfortunately, to do so optimally is an NP-hard optimization problem [14]. Hence, we settle on a simple greedy algorithm. For each edge not included in the tree, in order of decreasing edge weight, we add the edge to the graph if two conditions are met: first, we are able to easily verify that the treewidth stays less than $M$, and second, the length of the unique path in $\mathcal{S}_n$ between the endpoints is less than $L$. In order to bound the tree width, we maintain a counter at each node of the total number of added edges that result in a path through that node. Comparing to another method for constructing junction trees from spanning trees [15], one can check that the maximum node count is an upper-bound on the treewidth. We note that by using an appropriate directed representation of $\mathcal{S}_n$ relative to an arbitrary root, it is simple to identify the path between two nodes with complexity linear in path length $(< L)$.[4] Hence, the additional complexity of this greedy algorithm over that of the tree-selection procedure described previously is $\mathcal{O}(L|\mathcal{E}|)$.

In Figure 2 and Figure 3 we present a simple demonstration of the tree and hypertree selection procedures respectively, and the corresponding change in error achieved. The grayscale images represent the residual errors at the nodes of an $8 \times 8$ grid similar to $\mathcal{G}$ in Figure 1 (with white representing 1 and black representing 0), and the graphs beside them show the trees/hypertrees chosen based on these residual errors using the methods described above (the grid edge partial correlation coefficients are the same for all edges). Notice that the first tree in Figure 2 tries to include as many edges as possible that are incident on the nodes with high residual error. Such edges are useful for capturing walks ending at the high-error nodes, which contribute to the set of walks in (5). The first hypertree in Figure 3 actually includes *all* the edges incident on the high-error nodes. The residual errors after inference on these subgraphs are shown next in Figure 2 and Figure 3. As expected, the hypertree seems to achieve greater reduction in error compared to the spanning tree. Again, at this iteration, the subgraphs chosen by our methods adapt based on the errors at the various nodes.

## 5 Experimental illustration

### 5.1 Walk-summable models

We test the adaptive algorithms on densely connected nearest-neighbor grid-structured models (similar to $\mathcal{G}$ in Figure 1). We generate random grid models — the grid edge partial correlation coef-

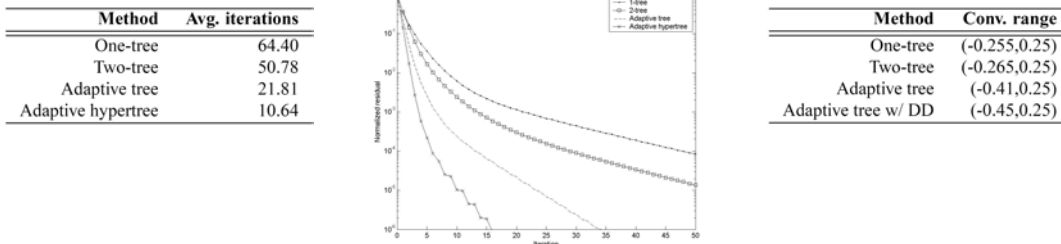

| Method | Avg. iterations |
|---|---|
| One-tree | 64.40 |
| Two-tree | 50.78 |
| Adaptive tree | 21.81 |
| Adaptive hypertree | 10.64 |

| Method | Conv. range |
|---|---|
| One-tree | (-0.255,0.25) |
| Two-tree | (-0.265,0.25) |
| Adaptive tree | (-0.41,0.25) |
| Adaptive tree w/ DD | (-0.45,0.25) |

Figure 4: (Left) Average number of iterations required for the normalized residual to reduce by a factor of $10^{-6}$ over 100 randomly generated $75 \times 75$ grid models; (Center) Convergence plot for a randomly generated $511 \times 511$ grid model; (Right) Convergence range in terms of partial correlation for 16-node cyclic model with edges to neighbors two steps away.

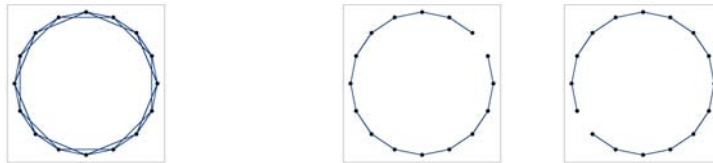

Figure 5: (Left) 16-node graphical model; (Right) two embedded spanning trees $\mathcal{T}_1$, $\mathcal{T}_2$.

ficients are chosen uniformly from $[-1, 1]$ and $R$ is scaled so that $\varrho(\bar{R}) = 0.99$. The vector $h$ is chosen to be the all-ones vector. The table on the left in Figure 4 shows the average number of iterations required by various algorithms to reduce the normalized residual error $\frac{\|h^{(n)}\|_2}{\|h^{(0)}\|_2}$ by a factor of $10^{-6}$. The average was computed based on 100 randomly generated $75 \times 75$ grid models. The plot in Figure 4 shows the decrease in the normalized residual error as a function of the number of iterations on a randomly generated $511 \times 511$ grid model. All these models are poorly conditioned because they are barely walk-summable (reason $\varrho(\bar{R}) = 0.99$). The stationary one-tree iteration uses a tree similar to $\mathcal{S}_1$ in Figure 1, and the two-tree iteration alternates between trees similar to $\mathcal{S}_1$ and $\mathcal{S}_3$ in Figure 1 [4]. The adaptive hypertree method uses $M = 6$ and $L = 8$. We also note that in practice the per-iteration costs of the adaptive tree and hypertree algorithms are roughly comparable.

These results show that our adaptive algorithms demonstrate significantly superior convergence properties compared to stationary methods, thus providing a convergent, computationally attractive method for estimation in walk-summable models. Our methods are applicable beyond Gaussian estimation to other problems that require solution of linear systems based on sparse, symmetric, positive-definite matrices. Several recent papers that develop machine learning algorithms are based on solving such systems of equations [8, 9]; in fact, both of these papers involve linear systems based on diagonally-dominant matrices, which are walk-summable.

## 5.2 Non-walk-summable models

Next, we give empirical evidence that our adaptive methods provide convergence over a broader range of models than stationary iterations. One potential complication in non-walk-summable models is that the subgraph models chosen by the stationary and adaptive algorithms may be indefinite or singular even though $J$ is positive-definite. In order to avoid this problem in the adaptive ET algorithm, the trees $\mathcal{S}_n$ chosen at each iteration must be valid (i.e., have positive-definite $J_{\mathcal{S}_n}$). A simple modification to the maximum-weight spanning tree algorithm achieves this goal — we add an extra condition to the algorithm to test for diagonal dominance of the chosen tree model (as all symmetric, diagonally-dominant models are positive definite [6]). That is, at each step of the maximum-weight spanning tree algorithm, we only add an edge if it does not create a cycle *and* maintains a diagonally-dominant tractable subgraph model. Consider the 16-node model on the left in Figure 5. Let all the edge partial correlation coefficients be $r$. The range of $r$ for which $J$ is positive-definite is roughly $(-0.46, 0.25)$, and the range for which the model is walk-summable is $(-0.25, 0.25)$ (in this range all the algorithms, both stationary and adaptive, converge). For the one-tree iteration we use tree $\mathcal{T}_1$, and for the two-tree iteration we alternate between trees $\mathcal{T}_1$ and $\mathcal{T}_2$ (see

Figure 5). As the table on the right in Figure 4 demonstrates, the adaptive tree algorithm without the diagonal-dominance (DD) check provides convergence over a much broader range of models than the one-tree and two-tree iterations, but not for all valid models. However, the modified adaptive tree algorithm with the DD check appears to converge almost up to the validity threshold. We have also observed such behavior empirically in many other (though not all) non-walk-summable models where the adaptive ET algorithm with the DD condition converges while stationary methods diverge. Thus, our adaptive methods, compared to stationary iterations, not only provide faster convergence rates in walk-summable models but also converge for a broader class of models.

## 6   Discussion

We analyze non-stationary iterations of the ET algorithm that use any sequence of subgraphs for estimation in Gaussian graphical models. Our analysis is based on the recently developed walk-sum interpretation of inference in Gaussian models, and we show that the ET algorithm converges for *any* sequence of subgraphs used in walk-summable models. These convergence results motivate the development of methods to choose subgraphs adaptively at each iteration to achieve maximum reduction in error. The adaptive procedures are based on walk-sum calculations, and minimize an upper bound on the error at each iteration. Our simulation results show that the adaptive algorithms provide a significant speedup in convergence over stationary methods. Moreover, these adaptive methods also seem to provide convergence over a broader class of models than stationary algorithms.

Our adaptive algorithms are greedy in that they only choose the "next-best" subgraph. An interesting question is to develop tractable methods to compute the next $K$ best subgraphs *jointly* to achieve maximum reduction in error *after $K$* iterations. The experiment with non-walk-summable models suggests that walk-sum analysis could be useful to provide convergent algorithms for *non-walk-summable* models, perhaps with restrictions on the order in which walk-sums are computed. Finally, subgraph preconditioners have been shown to improve the convergence rate of the conjugate-gradient method; using walk-sum analysis to select such preconditioners is of clear interest.

## Footnotes

[1]This can be achieved by performing the transformation $\tilde{J} \leftarrow D^{-\frac{1}{2}} J D^{-\frac{1}{2}}$, where $D$ is a diagonal matrix containing the diagonal entries of $J$.

[2] We generally denote the walk-sum of the set $\mathcal{W}(\sim)$ by $\phi(\sim)$.

[3] $K_{\mathcal{S}}$ can have non-zero diagonal in general, but we only consider the zero diagonal case here.

[4]One sets two pointers into the tree starting from any two nodes and then iteratively walks up the tree, always advancing from the point that is deeper in the tree, until the nearest ancestor of the two nodes is reached.

## References

[1] M. Luettgen, W. Carl, and A. Willsky. Efficient multiscale regularization with application to optical flow. *IEEE Transactions on Image Processing*, 3(1):41–64, Jan. 1994.

[2] P. Rusmevichientong and B. Van Roy. An Analysis of Turbo Decoding with Gaussian densities. In *Advances in Neural Information Processing Systems 12*, 2000.

[3] J. Pearl. *Probabilistic Reasoning in Intelligent Systems*. Morgan Kauffman, San Mateo, CA, 1988.

[4] E. Sudderth, M. Wainwright, and A. Willsky. Embedded Trees: Estimation of Gaussian processes on graphs with cycles. *IEEE Transactions on Signal Processing*, 52(11):3136–3150, Nov. 2004.

[5] D. Malioutov, J. Johnson, and A. Willsky. Walk-Sums and Belief Propagation in Gaussian Graphical Models. *Journal of Machine Learning Research*, 7:2031–2064, Oct. 2006.

[6] R. Varga. *Matrix Iterative Analysis*. Springer-Verlag, New York, 2000.

[7] R. Bru, F. Pedroche, and D. Szyld. Overlapping Additive and Multiplicative Schwarz iterations for H-matrices. *Linear Algebra and its Applications*, 393:91–105, Dec. 2004.

[8] D. Zhou, J. Huang, and B. Scholkopf. Learning from Labeled and Unlabeled data on a directed graph. In *Proceedings of the 22nd International Conference on Machine Learning*, 2005.

[9] D. Zhou and C. Burges. Spectral Clustering and Transductive Learning with multiple views. In *Proceedings of the 24th International Conference on Machine Learning*, 2007.

[10] V. Chandrasekaran, J. Johnson, and A. Willsky. Estimation in Gaussian Graphical Models using Tractable Subgraphs: A Walk-Sum Analysis. *To appear in IEEE Transactions on Signal Processing*.

[11] D. Malioutov, J. Johnson, and A. Willsky. GMRF variance approximation using spliced wavelet bases. In *IEEE International Conference on Acoustics, Speech and Signal Processing*, 2007.

[12] R. Godement. *Analysis I: Convergence, Elementary Functions*. Springer-Verlag, New York, 2004.

[13] T. Cormen, C. Leiserson, R. Rivest, and C. Stein. *Introduction to Algorithms*. MIT Press, 2001.

[14] N. Srebro. Maximum Likelihood Markov Networks: An Algorithmic Approach. Master's thesis, Massachusetts Institute of Technology, 2000.

[15] F. Kschischang, B. Frey, and H. Loeliger. Factor graphs and the sum-product algorithm. *IEEE Transactions on Information Theory*, 47(2):498–519, Feb. 2001.

